# Using hippocampal 'place cells' for navigation, exploiting phase coding

Neil Burgess, John O'Keefe and Michael Recce
Department of Anatomy, University College London,
London WC1E 6BT, England.
(e-mail: n.burgess@ucl.ac.uk)

## Abstract

A model of the hippocampus as a central element in rat navigation is presented. Simulations show both the behaviour of single cells and the resultant navigation of the rat. These are compared with single unit recordings and behavioural data. The firing of CA1 place cells is simulated as the (artificial) rat moves in an environment. This is the input for a neuronal network whose output, at each theta ($\theta$) cycle, is the next direction of travel for the rat. Cells are characterised by the number of spikes fired and the time of firing with respect to hippocampal $\theta$ rhythm. 'Learning' occurs in 'on-off' synapses that are switched on by simultaneous pre- and post-synaptic activity. The simulated rat navigates successfully to goals encountered one or more times during exploration in open fields. One minute of random exploration of a $1m^2$ environment allows navigation to a newly-presented goal from novel starting positions. A limited number of obstacles can be successfully avoided.

## 1  Background

Experiments have shown the hippocampus to be crucial to the spatial memory and navigational ability of the rat (O'Keefe & Nadel, 1978). Single unit recordings in freely moving rats have revealed 'place cells' in fields CA3 and CA1 of the hippocampus whose firing is restricted to small portions of the rat's environment (the corresponding 'place fields') (O'Keefe & Dostrovsky, 1971), see Fig. 1a. In addition cells have been found in the dorsal pre-subiculum whose primary behavioural

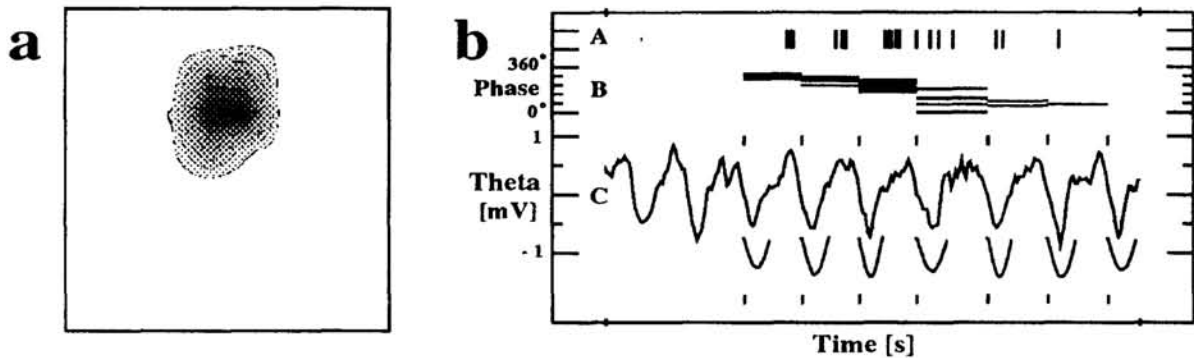

Figure 1: a) A typical CA1 place field, max. rate (over $1s$) is 13.6 spikes/s. b) One second of the EEG $\theta$ rhythm is shown in C, as the rat runs through a place field. A shows the times of firing of the place cell. Vertical ticks immediately above and below the EEG mark the positive to negative zero-crossings of the EEG, which we define as $0°$ (or $360°$) of phase. B shows the phase of $\theta$ at which each spike was fired (O'Keefe & Recce, 1992).

correlate is 'head-direction' (Taube et al., 1990). Both are suggestive of navigation.

Temporal as well as spatial aspects of the electrophysiology of the hippocampal region are significant for a model. The hippocampal EEG '$\theta$ rhythm' is best characterised as a sinusoid of frequency $7-12Hz$ and occurs whenever the rat is making displacement movements. Recently place cell firing has been found to have a systematic phase relationship to the local EEG (O'Keefe & Recce, 1992), see §3.1 and Fig. 1b. Finally, the $\theta$ rhythm has been found to modulate long-term potentiation of synapses in the hippocampus (Pavlides et al., 1988).

## 2  Introduction

We are designing a model that is consistent with both the data from single unit recording and the behavioural data that are relevant to spatial memory and navigation in the rat. As a first step this paper examines a simple navigational strategy that could be implemented in a physiologically plausible way to enable navigation to previously encountered reward sites from novel starting positions. We assume the firing properties of CA1 place cells, which form the input for our system.

The simplest map-based strategies (as opposed to route-following ones) are based on defining a surface over the whole environment, on which gradient ascent leads to the goal (e.g. delayed reinforcement or temporal difference learning). These tend to have the problem that, to build up this surface, the goal must be reached many times, from different points in the environment (by which time the rat has died of old age). Further, a new surface must be computed if the goal is moved. Specific problems are raised by the properties of rats' navigation: (i) the position of CA1 place fields is independent of goal position (Speakman & O'Keefe, 1990); (ii) high firing rates in place cells are restricted to limited portions of the environment; (iii) rats are able to navigate after a brief exploration of the environment, and (iv) can take novel short-cuts or detours (Tolman, 1948).

To overcome these problems we propose that a more diffuse representation of position is rapidly built up downstream of CA1, by cells with larger firing fields than in CA1. The patterns of activation of this group of cells, at two different locations in the environment, have a correlation that decreases with the separation of the two locations (but never reaches zero, as is the case with small place fields). Thus the overlap between the pattern of activity at any moment and the pattern of activity at the goal location would be a measure of nearness to the goal. We refer to these cells as 'subicular' cells because the subiculum seems a likely site for them, given single unit recordings (Barnes et al., 1990) showing spatially consistent firing over large parts of the environment.

We show that the output of these subicular cells is sufficient to enable navigation in our model. In addition the model requires: (i) 'goal' cells (see Fig. 4a) that fire when a goal is encountered, allowing synaptic connections from subicular cells to be switched on, (ii) phase-coded place cell firing, (iii) 'head-direction' cells, and (iv) synaptic change that is modulated by the phase of the EEG. The relative firing rates of groups of goal cells code for the direction of objects encountered during exploration, in the same way that cells in primate motor cortex code for the direction of arm movements (Georgopoulos et al., 1988).

## 3   The model

In our simulation a rat is in constant motion (speed $30cm/s$) in a square environment of size $L \times L$ ($L \leq 150cm$). Food or obstacles can be placed in the environment at any time. The rat is aware of any objects within $6cm$ (whisker length) of its position. It bounces off any obstacles (or the edge of the environment) with which it collides. The $\theta$ frequency is taken to be $10Hz$ (period $0.1s$) and we model each $\theta$ cycle as having 5 different phases. Thus the smallest timestep (at which synaptic connections and cell firing rates are updated) is $0.02s$. The rat is either 'exploring' (its current direction is a random variable within $30°$ of its previous direction), or 'searching' (its current direction is determined by the goal cells, see below). Synaptic and cell update rules are the same during searching or exploring.

### 3.1   The phase of CA1 place cell firing

When a rat on a linear track runs through a place field, the place cell fires at successively earlier phases of the EEG $\theta$ rhythm. A cell that fires at phase $360°$ when the rat enters the place field may fire as much as $355°$ earlier in the $\theta$ cycle when exiting the field (O'Keefe & Recce, 1992), see Fig. 1b.

Simulations below involve 484 CA1 place cells with place field centres spread evenly on a grid over the whole environment. The place fields are circular, with diameters $0.25L$, $0.35L$ or $0.4L$ (as place fields appear to scale with the size of an environment; Muller & Kubie, 1987). The fraction of cells active during any $0.1s$ interval is thus $\pi(0.125^2 + 0.175^2 + 0.2^2)/3 = 9\%$. When the rat is in a cell's place field it fires 1 to 3 spikes depending on its distance from the field centre, see Fig. 2b.

When the (simulated) rat first enters a place field the cell fires 1 spike at phase $360°$ of the $\theta$ rhythm; as the rat moves through the place field, its phase of firing shifts backwards by $72°$ every time the number of spikes fired by the cell changes

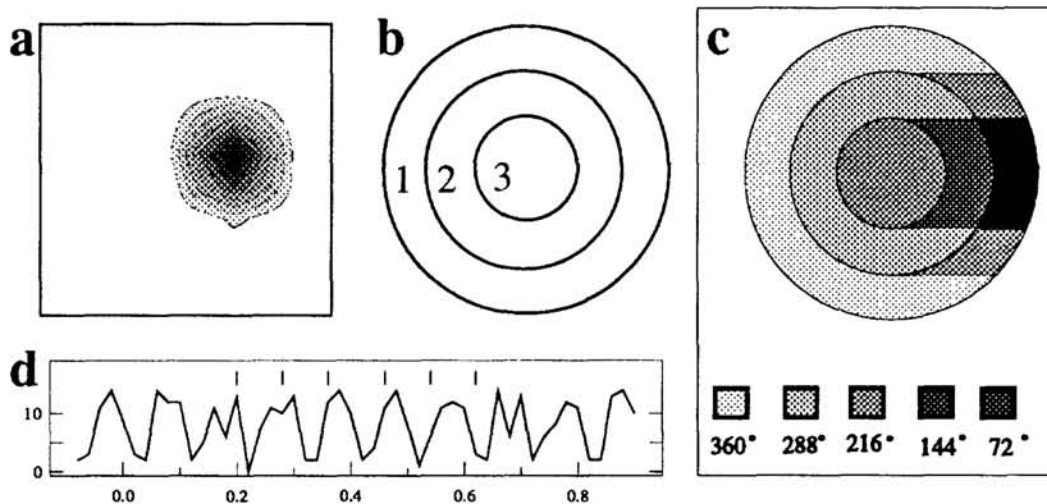

Figure 2: a) Firing rate map of a typical place cell in the model (max. rate 11.6 spikes/s); b) Model of a place field; the numbers indicate the number of spikes fired by the place cell when the rat is in each ring. c) The phase at which spikes would be fired during all possible straight trajectories of the rat through the place field *from left to right.* d) The total number of spikes fired in the model of CA1 versus time, the phase of firing of one place cell (as the rat runs through the centre of the field) is indicated be vertical ticks above the graph.

(i.e. each time it crosses a line in Fig. 2b). Thus each theta cycle is divided into 5 timesteps. No shift results from passing through the edge of the field, whereas a shift of 288° (0.08s) results from passing through the middle of the field, see Fig. 2c. The consequences for the model in terms of which place cells fire at different phases within one $\theta$ cycle are shown in Fig. 3. The cells that are active at phase 360° have place fields centred ahead of the position of the rat (i.e. place fields that the rat is entering), those active at phase 0° have place fields centred behind the rat. If the rat is simultaneously leaving field A and entering field B then cell A fires before cell B, having shifted backwards by up to 0.08s. The total number of spikes fired at each phase as the rat moves about implies that the envelope of all the spikes fired in CA1 oscillates with the $\theta$ frequency. Fig. 2d shows the shift in the firing of one cell compared to the envelope (cf. Fig. 1b).

## 3.2   Subicular cells

We simulate 6 groups of 80 cells (480 in total); each subicular cell receives one synaptic connection from a random 5% of the CA1 cells. These connections are either on or off (1 or 0). At each timestep (0.02s) the 10 cells in each group with the greatest excitatory input from CA1 fire between 1 and 5 spikes (depending on their relative excitation). Fig. 3c shows a typical subicular firing rate map. The consequences of phase coding in CA1 (Figs. 3a and b) remain in these subicular cells as they are driven by CA1: the net firing field of all cells active at phase 360° of $\theta$ is peaked *ahead* of the rat.

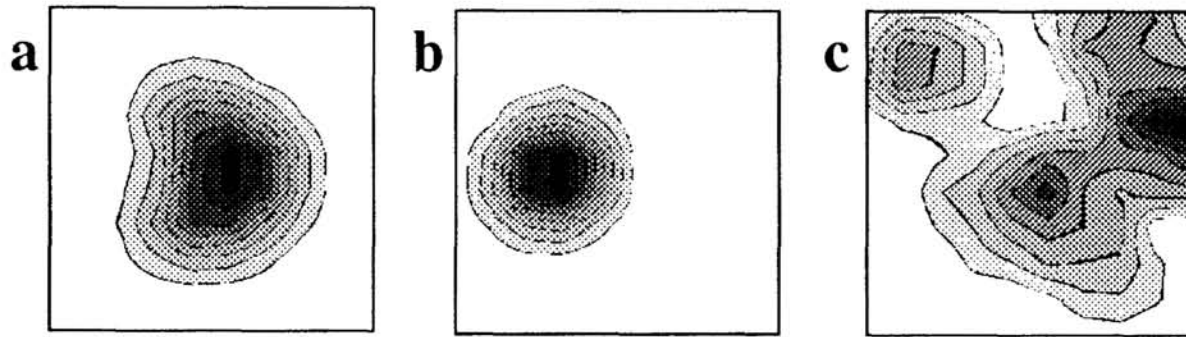

Figure 3: Net firing rate map of all the place cells that were active at the 360° (a) and 72° (b) phases of $\theta$ as the rat ran through the centre of the environment from left to right. c) Firing rate map of a typical 'subicular' cell in the model; max. rate (over $1.0s$) is 46.4 spikes/s. Barnes et al. (1990) found max. firing rates (over $0.1s$) of 80 spikes/s (mean 7 spikes/s) in the subiculum.

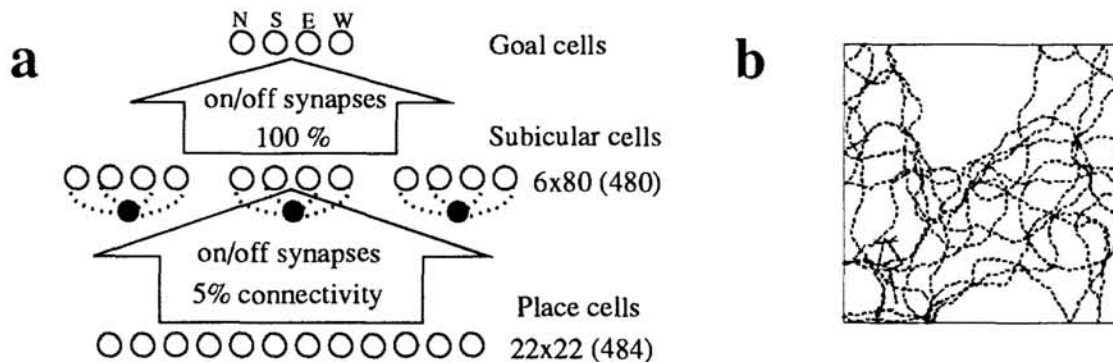

Figure 4: a) Connections and units in the model; interneurons shown between the subicular cells indicate competitive dynamics, but are not simulated explicitly. b) The trajectory of 90 seconds of 'exploration' in the central $126 \times 126cm^2$ of the environment. The rat is shown in the bottom left hand corner, to scale.

### 3.2.1  Learning

The connections are initialised such that each subicular cell receives on average one 'on' connection. Subsequently a synaptic connection can be switched on only during phases 180° to 360° of $\theta$. A synapse becomes switched on if the pre-synaptic cell is active, and the post-synaptic cell is above a threshold activity (4 spikes), in the same timestep ($0.02s$). Hence a subicular firing field is rapidly built up during exploration, as a superposition of CA1 place fields, see Fig 3c.

### 3.3  Goal cells

The correlation between the patterns of activity of the subicular cells at two different locations in the environment decreases with the separation of the two locations. Thus if synaptic connections to a goal cell were switched on when the rat encountered food then a firing rate map of the goal cell would resemble a cone covering the entire environment, peaked at the food site, i.e. the firing rate would indicate

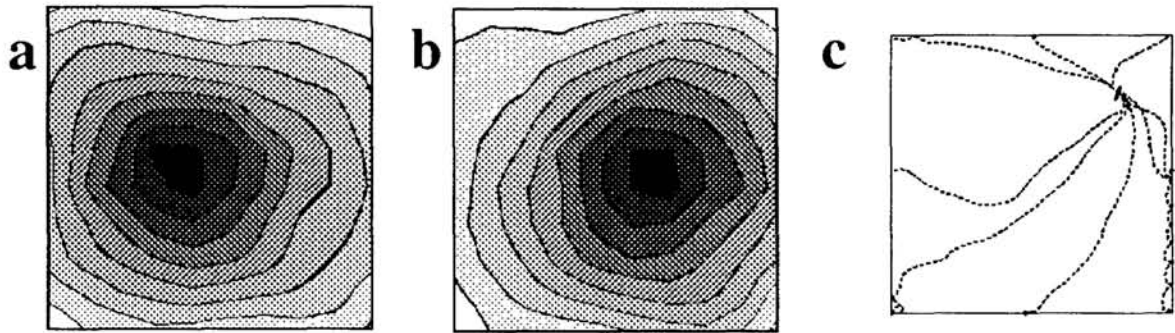

Figure 5: Goal cell firing fields, a) West, b) East, of 'food' encountered at the centre of the environment. c) Trajectories to a goal from 8 novel starting positions. All figures refer to encountering food immediately after the exploration in Fig. 4b. Notice that much of the environment was never visited during exploration.

the closeness of the food during subsequent movement of the rat. The scheme we actually use involves groups of goal cells continuously estimating the distance to 4 points displaced from the goal site in 4 different directions.

Notice that when a freely moving rat encounters an interesting object a fair amount of 'local investigation' takes place (sniffing, rearing, looking around and local exploration). During the local investigation of a small object the rat crosses the location of the object in many different directions. We postulate groups of goal cells that become excited strongly enough to induce synaptic change in connections from subicular cells whenever the rat encounters a specific piece of food and is heading in a particular direction. This supposes the joint action of an object classifier and of head-direction cells; head-direction cells corresponding to different directions being connected to different goal cells. Since synaptic change occurs only at the $180°$ to $360°$ phases of $\theta$, and the net firing rate map of all the subicular cells that are active at phase $360°$ during any $\theta$ cycle is peaked ahead of the rat, goal cells have firing fields that are peaked a little bit away from the goal position. For example, goal cells whose subicular connections are changed when the rat is heading east have firing rate fields that are peaked to the east of the goal location, see Fig. 5.

Local investigation of a food site is modelled by the rat moving $12cm$ to the north, south, east and west and occurs whenever food is encountered. Navigation is restricted to the central $126 \times 126cm^2$ portion of the $150 \times 150cm^2$ environment (over which firing rate maps are shown) to leave room for this. There are 4 goal cells for every piece of food found in the environment, (GC_north, GC_south, GC_east, GC_west), see Fig. 4a. Initially the connections from all subicular cells are off; they are switched on if the subicular cell is active and the rat is at the particular piece of food, travelling in the right direction. When the rat is searching, goal cells simply fire a number of spikes (in each $0.02s$ timestep) that is proportional to their net excitatory input from the subicular cells.

## 3.4   Maps and navigation

When the rat is to the north of the food, GC_north fires at a higher rate than GC_south. We take the firing rate of GC_north to be a 'vote' that the rat is north

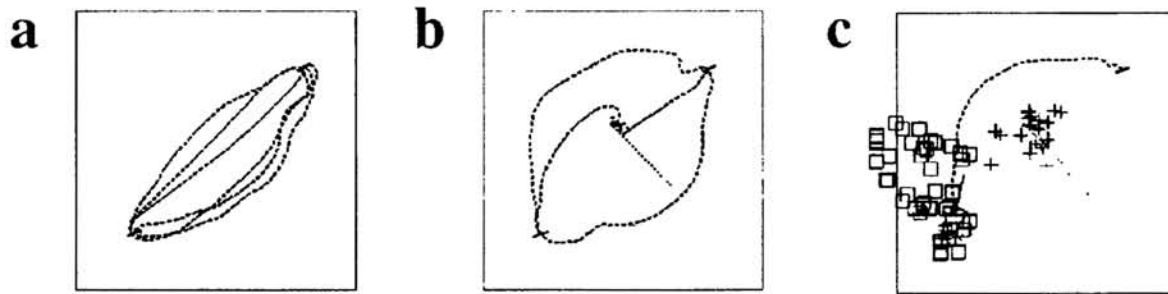

Figure 6: a) Trajectory of rat with alternating goals. b) an obstacle is interposed; the rat collides with the obstacle on the first run, but learns to avoid the collision site in the 2 subsequent runs. c) Successive predictions of goal (box) and obstacle (cross) positions generated as the rat ran from one goal site to the other; the predicted positions get more accurate as the rat gets closer to the object in question.

of the goal. Similarly the firing rate of GC_south is a vote that the rat is south of the goal: the resultant direction (the vector sum of directions north, south, east and west, weighted by the firing rates of the corresponding cells) is an estimate of the direction of the rat from the food (cf.Georgopoulos et al., 1988). Since the firing rate maps of the 4 goal cells are peaked quite close to the food location, their net firing rate increases as the food is approached, i.e. it is an estimation of how close the food is. Thus the firing rates of the 4 goal cells associated with a piece of food can be used to predict its approximate position relative to the rat (e.g. $70cm$ northeast), as the rat moves about the environment (see Fig. 6c).

We use groups of goal cells to code for the locations at which the rat encountered any objects (obstacles or food), as described above. A new group of goal cells is recruited every time the rat encounters a new object, or a new ($6cm$) part of an extended object. The output of the system acts as a *map* for the rat, telling it where everything is relative to itself, as it moves around. The process of *navigation* is to decide which way to go, given the information in the map. When there are no obstacles in the environment, navigation corresponds to moving in the direction indicated by the group of goal cells corresponding to a particular piece of food. When the environment includes many obstacles the task of navigation is much harder, and there is not enough clear behavioural data to guide modelling.

We do not model navigation at a neuronal level, although we wish to examine the navigation that would result from a simple reading of the 'map' provided by our model. The rules used to direct the simulated rat are as follows: (i) every $0.1s$ the direction and distance to the goal (one of the pieces of food) are estimated; (ii) the direction and distance to all locations at which an obstacle was encountered are estimated; (iii) obstacle locations are classified as 'in-the-way' if (a) estimated to be within 45° of the goal direction, (b) closer than the goal and (c) less than $L/2$ away; (iv) the current direction of the rat becomes the vector sum of the goal direction (weighted by the net firing rate of the corresponding 4 goal cells) minus the directions to any in-the-way obstacles (weighted by the net firing rate of the 'obstacle cells' and by the similarity of the obstacle and goal directions).

## 4  Performance

The model achieves latent learning (i.e. the map is constructed independently of knowledge of the goal, see e.g. Tolman, 1948). A piece of food encountered only once, after exploration, can be returned to, see Fig. 5c. Notice that a large part of the environment was never visited during exploration (Fig. 4b). Navigation is equally good after exploration in an environment containing food/obstacles from the beginning. If the food is encountered only during the earliest stages of exploration (before a stable subicular representation is built up) then performance is worse. Multiple goals and a small number of obstacles can be accommodated, see Fig. 6. Notice that searching also acts as exploration, and that synaptic connections can be switched at any time: all learning is incremental, but saturates when all the relevant synapses have been switched on. Performance does not depend crucially on the parameter values, used although it is worse with fewer cells, and smaller environments require less exploration before reliable navigation is possible (e.g. $60s$ for a $1m^2$ box). Quantitative analysis will appear in a longer paper.

## References

Barnes C A, McNaughton B L, Mizumori S J Y, Leonard B W & Lin L-H (1990) 'Comparison of spatial and temporal characteristics of neuronal activity in sequential stages of hippocampal processing', *Progress in Brain Research* **83** 287-300.

Georgopoulos A P, Kettner R E & Schwartz A B (1988) 'Primate motor cortex and free arm movements to visual targets in three-dimensional space. II. Coding of the direction of movement by a neuronal population', *J. Neurosci.* **8** 2928-2937.

Muller R U & Kubie J L (1987) 'The effects of changes in the environment on the spatial firing of hippocampal complex-spike cells', *J. Neurosci.* **7** 1951-1968.

O'Keefe J & Dostrovsky J (1971) 'The hippocampus as a spatial map: preliminary evidence from unit activity in the freely moving rat', *BrainRes.* **34** 171-175.

O'Keefe J & Nadel L (1978) *The hippocampus as a cognitive map*, Clarendon Press, Oxford.

O'Keefe J & Recce M (1992) 'Phase relationship between hippocampal place units and the EEG theta rhythm', *Hippocampus,* to be published.

Pavlides C, Greenstein Y J, Grudman M & Winson J (1988) 'Long-term potentiation in the dentate gyrus is induced preferentially on the positive phase of $\theta$-rhythm', *Brain Res.* **439** 383-387.

Speakman A S & O'Keefe J (1990) 'Hippocampal complex spike cells do not change their place fields if the goal is moved within a cue controlled environment', *European Journal of Neuroscience* **2** 544-555.

Taube J S, Muller R U & Ranck J B Jr (1990) 'Head-direction cells recorded from the postsubiculum in freely moving rats. I. Description & quantitative analysis', *J. Neurosci.* **10** 420-435.

Tolman E C (1948) 'Cognitive Maps in rats and men', *Psychological Review* **55** 189-208.
